# Learning about Multiple Objects in Images: Factorial Learning without Factorial Search

**Christopher K. I. Williams**[*]   **and Michalis K. Titsias**
School of Informatics, University of Edinburgh, Edinburgh EH1 2QL, UK
c.k.i.williams@ed.ac.uk      M.Titsias@sms.ed.ac.uk

## Abstract

We consider data which are images containing views of multiple objects. Our task is to learn about each of the objects present in the images. This task can be approached as a factorial learning problem, where each image must be explained by instantiating a model for each of the objects present with the correct instantiation parameters. A major problem with learning a factorial model is that as the number of objects increases, there is a combinatorial explosion of the number of configurations that need to be considered. We develop a method to extract object models sequentially from the data by making use of a robust statistical method, thus avoiding the combinatorial explosion, and present results showing successful extraction of objects from real images.

## 1 Introduction

In this paper we consider data which are images containing views of multiple objects. Our task is to learn about each of the objects present in the images. Previous approaches (discussed in more detail below) have approached this as a factorial learning problem, where each image must be explained by instantiating a model for each of the objects present with the correct instantiation parameters. A serious concern with the factorial learning problem is that as the number of objects increases, there is a combinatorial explosion of the number of configurations that need to be considered. Suppose there are $L$ possible objects, and that there are $J$ possible values that the instantiation parameters of any one object can take on; we will need to consider $O(J^L)$ combinations to explain any image. In contrast, in our approach we find one object at a time, thus avoiding the combinatorial explosion.

In unsupervised learning we aim to identify regularities in data such as images. One fairly simple unsupervised learning model is clustering, which can be viewed as a mixture model where there are a finite number of types of object, and data is produced by choosing *one* of these objects and then generating the data conditional on this choice. As a model of objects in images standard clustering approaches are limited as they do not take into account the variability that can arise due to the transformations that can take place, described by instantiation parameters such as translation, rotation *etc* of the object. Suppose that there are $m$ different instantiation parameters, then a single object will sweep out a $m$-dimensional manifold in the image space. Learning about objects taking this regularity into account has

---

[*]http://anc.ed.ac.uk

been called transformation-invariant clustering by Frey and Jojic (1999, 2002). However, this work is still limited to finding a single object in each image.

A more general model for data is that where the observations are explained by multiple causes; in our example this will be that in each image there are $L$ objects. The approach of Frey and Jojic (1999, 2002) can be extended to this case by explicitly considering the simultaneous instantiation of all $L$ objects (Jojic and Frey, 2001). However, this gives rise to a large search problem over the instantiation parameters of all objects simultaneously, and approximations such as variational methods are needed to carry out the inference. In our method, by contrast, we discover the objects one at a time using a robust statistical method. Sequential object discovery is possible because multiple objects combine by occluding each other.

The general problem of factorial learning has longer history, see, for example, Barlow (1989), Hinton and Zemel (1994), and Ghahramani (1995). However, Frey and Jojic made the important step for image analysis problems of using explicit transformations of object models, which allows the incorporation of prior knowledge about these transformations and leads to good interpretability of the results.

A related line of research is that concerned with discovering part decompositions of objects. Lee and Seung (1999) described a non-negative matrix factorization method addressing this problem, although their work does not deal with parts undergoing transformations. There is also work on learning parts by Shams and von der Malsburg (1999), which is compared and contrasted with our work in section 4.

The structure of the remainder of this paper is as follows. In section 2 we describe the model, first for images containing only a single object (§2.1) and then for images containing multiple objects (§2.2). In section 3 we present experimental results for up to five objects appearing against stationary and non-stationary backgrounds. We conclude with a discussion in section 4.

## 2 Theory

### 2.1 Learning one object

In this section we consider the problem of learning about one object which can appear at various locations in an image. The object is in the *foreground*, with a background behind it. This background can either be fixed for all training images, or vary from image to image. The two key issues that we must deal with are (i) the notion of a pixel being modelled as foreground or background, and (ii) the problem of transformations of the object. We consider first the foreground/background issue.

Consider an image $\mathbf{x}$ of size $P_x \times P_y$ containing $P \stackrel{def}{=} P_x P_y$ pixels, arranged as a length $P$ vector. Our aim is to learn appearance-based representations of the foreground $\mathbf{f}$ and the background $\mathbf{b}$. As the object will be smaller than $P_x \times P_y$ pixels, we will need to specify which pixels belong to the background and which to the foreground; this is achieved by a vector of binary latent variables $\mathbf{s}$, one for each pixel. Each binary variable in $\mathbf{s}$ is drawn independently from the corresponding entry in a vector of probabilities $\boldsymbol{\pi}$. For pixel $p$, if $\pi_p \simeq 0$, then the pixel will be ascribed to the background with high probability, and if $\pi_p \simeq 1$, it will be ascribed to the foreground with high probability. We sometimes refer to $\boldsymbol{\pi}$ as a mask.

$x_p$ is modelled by a mixture distribution:

$$x_p \sim \begin{cases} p_f(x_p; f_p) = N(x_p; f_p, \sigma_f^2) & \text{if } s_p = 1, \\ p_b(x_p; b_p) = N(x_p; b_p, \sigma_b^2) & \text{if } s_p = 0, \end{cases} \tag{1}$$

where $\sigma_f^2$ and $\sigma_b^2$ are respectively the foreground and background variances. Thus, ignoring transformations, we obtain $p(\mathbf{x}) = \prod_{p=1}^{P}[\pi_p p_f(x_p; f_p) + (1 - \pi_p)p_b(x_p; b_p)]$.

The second issue that we must deal with is that of transformations. Below we consider only translations, although the ideas can be extended to deal with other transformations such as scaling and rotation (see e.g. Jojic and Frey (2001)). Each possible transformation (e.g. translations in units of one pixel) is represented by a corresponding transformation matrix, so that matrix $T_j$ corresponds to transformation $j$ and $T_j\mathbf{f}$ is the transformed foreground model. In our implementation the translations use wrap-around, so that each $T_j$ is in fact a permutation matrix. The semantics of foreground and background mean that the mask $\boldsymbol{\pi}$ must also be transformed, so that we obtain

$$p(\mathbf{x}|T_j) = \prod_{p=1}^{P}[(T_j\boldsymbol{\pi})_p p_f(x_p; (T_j\mathbf{f})_p) + (1 - (T_j\boldsymbol{\pi})_p)p_b(x_p; b_p)]. \tag{2}$$

Notice that the foreground $\mathbf{f}$ and mask $\boldsymbol{\pi}$ are transformed by $T_j$, but the background $\mathbf{b}$ is not. In order for equation 2 to make sense, each element of $T_j\boldsymbol{\pi}$ must be a valid probability (lying in $[0, 1]$). This is certainly true for the case when $T_j$ is a permutation matrix (and can be true more generally).

To complete the model we place a prior probability $p_j$ on each transformation $T_j$; this is taken to be uniform over all possibilities so that $p(\mathbf{x}) = \sum_{j=1}^{J} p_j p(\mathbf{x}|T_j)$. Given a data set $\{\mathbf{x}^n\}$, $n = 1, \ldots, N$ we can adapt the parameters $\theta = (\mathbf{f}, \boldsymbol{\pi}, \mathbf{b}, \sigma_f^2, \sigma_b^2)$ by maximizing the log likelihood $L(\theta) = \sum_{n=1}^{N} \log p(\mathbf{x}^n|\theta)$. This can be achieved through using the EM algorithm to handle the missing data which is the transformation and $\mathbf{s}$.

The model developed in this section is similar to Jojic and Frey (2001), except that our mask $\boldsymbol{\pi}$ has probabilistic semantics, which means that an exact M-step can be used as opposed to the generalized M-step used by Jojic and Frey.

## 2.2 Coping with multiple objects

If there are $L$ foreground objects, one natural approach is to consider models with $L$ latent variables, each taking on the $J$ values of the possible transformations. We also need to account for object occlusions. By assuming that the $L$ objects can arbitrarily occlude one another (and this occlusion ordering can change in different images), there are $L!$ possible arrangements. A model that accounts for multiple objects is described in Jojic and Frey (2001) where the occlusion ordering of the objects is taken as being fixed since they assume that each object is ascribed to a global layer. A full search over the parameters (assuming unknown occlusion ordering for each image) must consider $J^L L!$ possibilities, which scales exponentially with $L$. An alternative is to consider approximations; Ghahramani (1995) suggests mean field and Gibbs sampling approximations and Jojic and Frey (2001) use approximate variational inference.

Our goal is to find one object at a time in the images. We describe two methods for doing this. The first uses random initializations, and on different runs can find different objects; we denote this RANDOM STARTS. The second method (denoted GREEDY) removes objects found in earlier iterations and looks for as-yet-undiscovered objects in what remains.

For both methods we need to adapt the model presented in section 2.1. The problem is that occlusion can occur of both the foreground and the background. For a foreground pixel, a different object to the one being modelled may be interposed between the camera and our object, thus perturbing the pixel value. This can be modelled with a mixture distribution as $p_f(x_p; f_p) = \alpha_f N(x_p; f_p, \sigma_f^2) + (1 - \alpha_f)U(x_p)$, where $\alpha_f$ is the fraction of times

a foreground pixel is not occluded and the robustifying component $U(x_p)$ is a uniform distribution common for all image pixels. Such robust models have been used for image matching tasks by a number of authors, notably Black and colleagues (Black and Jepson, 1996).

Similarly for the background, a different object from the one being modelled may be interposed between the background and the camera, so that we again have a mixture model $p_b(x_p; b_p) = \alpha_b N(x_p; b_p, \sigma_b^2) + (1 - \alpha_b)U(x_p)$, with similar semantics for the parameter $\alpha_b$. (If the background has high variability then this robustness may not be required, but it will be in the case that the background is fixed while the objects move.)

### 2.2.1 Finding the first object

With this robust model we can now apply the RANDOM STARTS algorithm by maximizing the likelihood of a set of images with respect to the model using the EM algorithm. The expected complete data log likelihood is given by

$$Q = \sum_{n=1}^{N}\sum_{j=1}^{J} P(T_j|\mathbf{x}^n)\{(\overline{\mathbf{s}}_j^n)^T (\log T_j \boldsymbol{\pi} + \mathbf{r}_f^{n,j} * (-\frac{1}{2\sigma_f^2}(\mathbf{x}^n - T_j\mathbf{f})^2 - \frac{1}{2}\log \sigma_f^2 \mathbf{1})$$

$$+ (\mathbf{1} - \overline{\mathbf{s}}_j^n)^T (\log(\mathbf{1} - T_j\boldsymbol{\pi}) + \mathbf{r}_b^n * (-\frac{1}{2\sigma_b^2}(\mathbf{x}^n - \mathbf{b})^2 - \frac{1}{2}\log \sigma_b^2 \mathbf{1})\} + const, \quad (3)$$

where $\mathbf{y} * \mathbf{z}$ defines the element-wise product between two vectors, $\mathbf{y} * \mathbf{y}$ is written as $\mathbf{y}^2$ for compactness and $\mathbf{1}$ denotes the $P$-dimensional vector containing ones. The expected values of several latent variables are as follows: $P(T_j|\mathbf{x}) = \frac{p_j p(\mathbf{x}|T_j)}{\sum_{i=1}^{M} p_i p(\mathbf{x}|T_i)}$ is the transformation responsibility, $\overline{\mathbf{s}}_j^n$ is a $P$-dimensional vector associated with the binary variables $\mathbf{s}$ with each element storing the probability $p(s_p = 1|\mathbf{x}^n, T_j) = \frac{(T_j\boldsymbol{\pi})_p p_f(x_p;(T_j\mathbf{f})_p)}{(T_j\boldsymbol{\pi})_p p_f(x_p;(T_j\mathbf{f})_p)+(1-(T_j\boldsymbol{\pi})_p)p_b(x_p;b_p)}$, $\mathbf{r}_f^{n,j}$ is the vector containing the robust responsibilities for the foreground on image $\mathbf{x}^n$ using transformation $T_j$, so that its $p^{th}$ element is equal to $\frac{\alpha_f N(x_p^n;(T_j\mathbf{f})_p,\sigma_f^2)}{\alpha_f N(x_p^n;(T_j\mathbf{f})_p,\sigma_f^2)+(1-\alpha_f)U(x_p^n)}$ and similarly the vector $\mathbf{r}_b^n$ defines the robust responsibilities of the background. Note that the latter responsibilities do not depend on the transformation $T_j$ since the background is not transformed.

All of the above expected values of the missing variables are estimated in the $E$-step using the current parameter values. In the $M$-step we maximise the $Q$ function with respect to the model parameters $\boldsymbol{\pi}, \mathbf{f}, \mathbf{b}, \sigma_f^2$ and $\sigma_b^2$. We do not have space to show all of the updates but for example

$$\mathbf{f} \leftarrow \sum_{n=1}^{N}\sum_{j=1}^{J} P(T_j|\mathbf{x}^n)T_j^T[\overline{\mathbf{s}}_j^n * \mathbf{r}_f^{n,j} * \mathbf{x}^n]./\sum_{n=1}^{N}\sum_{j=1}^{J} P(T_j|\mathbf{x}^n)T_j^T[\overline{\mathbf{s}}_j^n * \mathbf{r}_f^{n,j}], \quad (4)$$

where $\mathbf{y}./\mathbf{z}$ stands for the element-wise division between two vectors. This update is quite intuitive. Consider the case when $P(T_j|\mathbf{x}) = 1$ for $j = j^*$ and 0 otherwise. For pixels which are ascribed to the foreground (i.e. $(\overline{\mathbf{s}}_{j*}^n * \mathbf{r}_f^{n,j^*})_p \simeq 1$), the values in $\mathbf{x}^n$ are transformed by $T_{j*}^T$ (which is $T_{j*}^{-1}$ as the transformations are permutation matrices). This removes the effect of the transformation and thus allows the foreground pixels found in each training image to be averaged to produce $\mathbf{f}$.

On different runs we hope to discover objects. However, this is rather inefficient as the basins of attraction for the different objects may be very different in size given the initialization. Thus we describe the GREEDY algorithm next.

### 2.2.2 The GREEDY algorithm

We assume that we have run the RANDOM STARTS algorithm and have learned a fore-ground model $\mathbf{f}_1$ and mask $\boldsymbol{\pi}_1$. We wish to remove from consideration the pixels of the learned object (in each training image) in order to find a new object by applying the same algorithm. For each example image $\mathbf{x}$ we can use the responsibilities $p(T_j|\mathbf{x})$ to find the most likely transformation $i_1^*$.[1] Now note that the transformed mask $T_{i_1^*}\boldsymbol{\pi}_1$ obtains values close to 1 for all object pixels, however some of these pixels might be occluded by other not-yet-discovered objects and we do not wish to remove them from consideration. Thus we consider the vector $\boldsymbol{\rho}_1 = (T_{i_1^*}\boldsymbol{\pi}_1) * \mathbf{r}_{f_1}^{i_1^*}$. According to the semantics of the robust foreground responsibilities $\mathbf{r}_{f_1}^{i_1^*}$, $\boldsymbol{\rho}_1$ will roughly give close to 1 values only for the non-occluded object pixels. To further explain all pixels having $(\boldsymbol{\rho}_1)_p \simeq 0$ we introduce a new foreground model $\mathbf{f}_2$ and mask $\boldsymbol{\pi}_2$, then for each transformation $T_j$ of model 2, we obtain

$$p(\mathbf{x}|T_{i_1^*}, T_j) = \prod_{p=1}^{P} [(\boldsymbol{\rho}_1)_p N(x_p; (T_{i_1^*}\mathbf{f}_1)_p, \sigma_{f_1}^2) +$$

$$(1 - \boldsymbol{\rho}_1)_p \left( (T_j\boldsymbol{\pi}_2)_p p_f(x_p; (T_j\mathbf{f}_2)_p) + (1 - T_j\boldsymbol{\pi}_2)_p p_b(x_p; b_p) \right) ]. \quad (5)$$

Note that we have dropped the robustifying component $U(x_p)$ from model 1, since the parameters of this object have been learned. By summing out over the possible transforma-tions we can maximize the likelihood with respect to $\mathbf{f}_2$, $\boldsymbol{\pi}_2$, $\sigma_{f_2}^2$, $\mathbf{b}$ and $\sigma_b^2$.

The above expression says that each image pixel $x_p$ is modelled by a three-component mixture distribution; the pixel $x_p$ can belong to the first object with probability $(\boldsymbol{\rho}_1)_p$, does not belong to the first object and belongs to the second one with probability $(1 - \boldsymbol{\rho}_1)_p(T_j\boldsymbol{\pi}_2)_p$, while with the remaining probability it is background. Thus, the search for a new object involves only the pixels that are not accounted for by model 1 (i.e. those for which $(\boldsymbol{\rho}_1)_p \simeq 0$).

This process can be continued, so that after finding a second model, the remaining back-ground is searched for a third model, and so on. The formula for $L$ objects becomes

$$p(\mathbf{x}|T_{i_1^*}, \ldots, T_{i_{L-1}^*}, T_j) = \prod_{p=1}^{P} [\sum_{\ell=1}^{L-1} \prod_{k=1}^{\ell-1} (1 - \rho_k)_p (\rho_\ell)_p N(x_p; (T_{i_\ell^*}\mathbf{f}_\ell)_p, \sigma_{f_\ell}^2) +$$

$$\prod_{\ell=1}^{L-1} (1 - \rho_\ell)_p (T_j\boldsymbol{\pi}_L)_p p_f(x_p; (T_j\mathbf{f}_L)_p) + \prod_{\ell=1}^{L-1} (1 - \rho_\ell)_p (1 - T_j\boldsymbol{\pi}_L)_p p_b(x_p; b_p) ]. \quad (6)$$

This is a $L+1$ component mixture at each pixel, where the $L+1^{th}$ object is the background. If $\ell = 1$ then the term $\prod_{k=1}^{\ell-1}(1-\rho_\ell^{i_\ell^*})_p$ is defined to be equal to 1. Note that all parameters of the first $L-1$ components are kept fixed (learned in previous stages). We always deal with only one object at a time and thus with one transformation latent variable. This approach can be viewed as approximating the full factorial model by sequentially learning each factor (object). A crucial point is that the algorithm is not assumed to extract layers in images, ordered from the nearest layer to the furthest one. In fact in next section we show a two-object example of a video sequence where we learn first the occluded object.

Space limitations do not permit us to show the $Q$ function and updates for the parameters, but these are very similar to the RANDOM STARTS, since we also learn only the param-eters of one object plus the background while keeping fixed all the parameters of previously discovered objects.

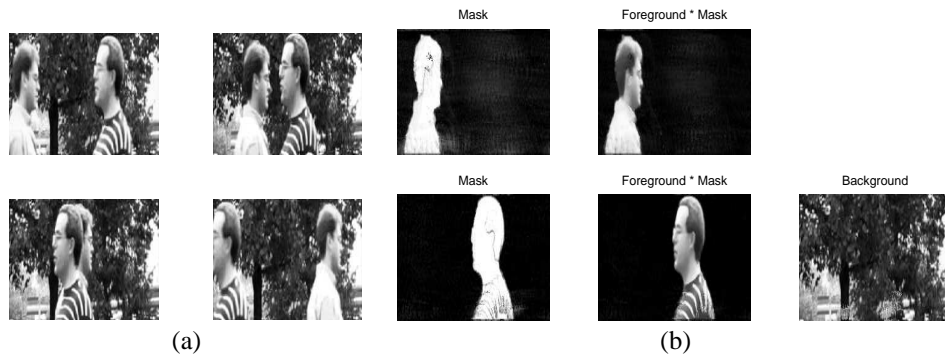

Figure 1: Learning two objects against a stationary background. Panel (a) displays some frames of the training images, and (b) shows the two objects and background found by the GREEDY algorithm.

## 3   Experiments

We describe three experiments extracting objects from images including up to five movable objects, using stationary as well as non-stationary backgrounds. In these experiments the uniform distribution $U(x_p)$ is based on the maximum and minimum pixel values of all training image pixels. In all the experiments reported below $\alpha_f$ and $\alpha_b$ were chosen to be 0.9. Also we assume that the total number of objects $L$ that appear in the images is known, thus the GREEDY algorithm terminates when we discover the $L^{th}$ object.

The learning algorithm also requires the initialization of the foreground $\mathbf{f}$ and background appearances $\mathbf{b}$, the mask $\boldsymbol{\pi}$ and the parameters $\sigma_f^2$ and $\sigma_b^2$. Each element of the mask $\boldsymbol{\pi}$ is initialised to 0.5, the background appearance $\mathbf{b}$ to the mean of the training images and the variances $\sigma_f^2$ and $\sigma_b^2$ are initialized to equal large values (larger than the overall variance of all image pixels). For the foreground appearance $\mathbf{f}$ we compute the pixelwise mean of the training images and add independent Gaussian noise with the equal variances at each pixel, where the variance is set to be large enough so that the range of pixel values found in the training images can be explored.

In the GREEDY algorithm each time we add a new object $\ell$ the parameters $\mathbf{f}_\ell$, $\mathbf{b}$, $\boldsymbol{\pi}_\ell$, $\sigma_{f_\ell}^2$, $\sigma_b^2$ are initialized as described above. This means that the background $\mathbf{b}$ is reset to the mean of the training images; this is done to avoid local maxima since the background found by considering only some of the objects in the images can be very different than the true background.

Figure 1 illustrates the detection of two objects against a stationary background[2]. Some examples of the 44 $118 \times 248$ training images (excluding the black border) are shown in Figure 1(a) and results are shown in Figure 1(b). For both objects we show both the learned mask and the elementwise product of the learned foreground and mask. In most runs the person with the lighter shirt (Jojic) is discovered first, even though he is occluded and the person with the striped shirt (Frey) is not. Video sequences of the raw data and the extracted objects can be viewed at `http://www.dai.ed.ac.uk/homes/s0129556/lmo.html`.

In Figure 2 five objects are learned against a stationary background, using a dataset of 80 images of size $66 \times 88$. Notice the large amount of occlusion in some of the training images shown in Figure 2(a). Results are shown in Figure 2(b) for the GREEDY algorithm.

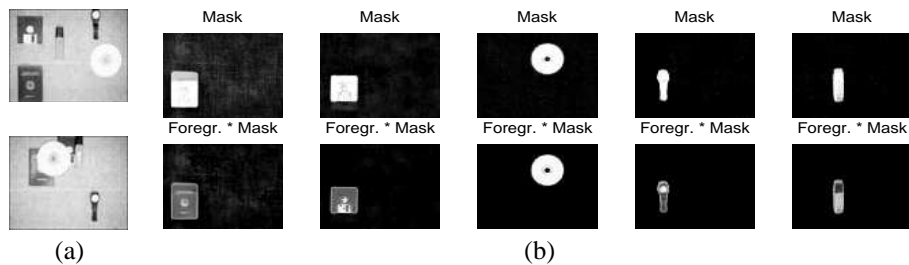

Figure 2: Learning five objects against a stationary background. Panel (a) displays some of the training images and (b) shows the objects learned by the GREEDY algorithm.

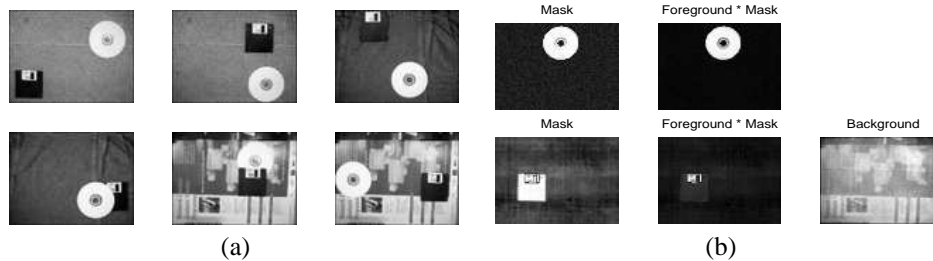

Figure 3: Two objects are learned from a set of images with non-stationary background. Panel (a) displays some examples of the training images, and (b) shows the objects found by the GREEDY algorithm.

In Figure 3 we consider learning objects against a non-stationary background. Actually three different backgrounds were used, as can be seen in the example images shown in Figure 3(a). There were 67 $66 \times 88$ images in the training set. Using the RANDOM STARTS algorithm the CD was found in 9 out of 10 runs. The results with the GREEDY algorithm are shown in Figure 3(b). The background found is approximately the average of the three backgrounds.

Overall we conclude that the RANDOM STARTS algorithm is not very effective at finding multiple objects in images; it needs many runs from different initial conditions, and sometimes fails entirely to find all objects. In contrast the GREEDY algorithm is very effective.

## 4   Discussion

Shams and von der Malsburg (1999) obtained candidate parts by matching images in a pairwise fashion, trying to identify corresponding regions in the two images. These candidate image patches were then clustered to compensate for the effect of occlusions. We make four observations: (i) instead of directly learning the models, they match each image against all others (with complexity $O(N^2)$), as compared to the linear scaling with $N$ in our method; (ii) in their method the background must be removed otherwise it would give rise to large match regions; (iii) they do not define a probabilistic model for the images (with all its attendant benefits); (iv) their data (although based on realistic CAD-type models) is synthetic, and designed to focus learning on shape related features by eliminating complicating factors such as background, surface markings etc.

In our work the model for each pixel is a mixture of Gaussians. There is some previous

work on pixelwise mixtures of Gaussians (see, e.g. Rowe and Blake 1995) which can, for example, be used to achieve background subtraction and highlight moving objects against a stationary background. Our work extends beyond this by gathering the foreground pixels into objects, and also allows us to learn objects in the more difficult non-stationary background case. For the stationary background case, pixelwise mixture of Gaussians might be useful ways to create candidate objects.

The GREEDY algorithm has shown itself to be an effective factorial learning algorithm for image data. We are currently investigating issues such as dealing with richer classes of transformations, detecting $L$ automatically, and allowing objects not to appear in all images. Furthermore, although we have described this work in relation to image modelling, it can be applied to other domains. For example, one can make a model for sequence data by having Hidden Markov models (HMMs) for a "foreground" pattern and the "background". Faced with sequences containing multiple foreground patterns, one could extract these patterns sequentially using a similar algorithm to that described above. It is true that for sequence data it would be possible to train a compound HMM consisting of $L+1$ HMM components simultaneously, but there may be severe local minima problems in the search space so that the sequential approach might be preferable.

Acknowledgements: CW thanks Geoff Hinton for helpful discussions concerning the idea of learning one object at a time.

## Footnotes

[1]It would be possible to make a "softer" version of this, where the transformations are weighted by their posterior probabilities, but in practice we have found that these probabilities are usually 1 for the best-fitting transformation and 0 otherwise after learning $\mathbf{f}_1$ and $\boldsymbol{\pi}_1$.

[2]These data are used in Jojic and Frey (2001). We thank N. Jojic and B. Frey for making available these data via `http://www.psi.toronto.edu/layers.html`.

## References

Barlow, H. (1989). Unsupervised Learning. *Neural Computation*, 1:295–311.

Black, M. J. and Jepson, A. (1996). EigenTracking: Robust matching and tracking of articulated objects using a view-based representation. In Buxton, B. and Cipolla, R., editors, *Proceedings of the Fourth European Conference on Computer Vision, ECCV'96*, pages 329–342. Springer-Verlag.

Frey, B. J. and Jojic, N. (1999). Estimating mixture models of images and inferring spatial transformations using the EM algorithm. In *Proceedings of the IEEE Conference on Computer Vision and Pattern Recognition 1999*. IEEE Computer Society Press. Ft. Collins, CO.

Frey, B. J. and Jojic, N. (2002). Transformation Invariant Clustering and Linear Component Analysis Using the EM Algorithm. Revised manuscript under review for IEEE PAMI.

Ghahramani, Z. (1995). Factorial Learning and the EM Algorithm. In Tesauro, G., Touretzky, D. S., and Leen, T. K., editors, *Advances in Neural Information Processing Systems 7*, pages 617–624. Morgan Kaufmann, San Mateo, CA.

Hinton, G. E. and Zemel, R. S. (1994). Autoencoders, minimum description length, and Helmholtz free energy. In Cowan, J., Tesauro, G., and Alspector, J., editors, *Advances in Neural Information Processing Systems 6*. Morgan Kaufmann.

Jojic, N. and Frey, B. J. (2001). Learning Flexible Sprites in Video Layers. In *Proceedings of the IEEE Conference on Computer Vision and Pattern Recognition 2001*. IEEE Computer Society Press. Kauai, Hawaii.

Lee, D. D. and Seung, H. S. (1999). Learning the parts of objects by non-negative matrix factorization. *Nature*, 401:788–791.

Rowe, S. and Blake, A. (1995). Statistical Background Modelling For Tracking With A Virtual Camera. In Pycock, D., editor, *Proceedings of the 6th British Machine Vision Conference*, volume volume 2, pages 423–432. BMVA Press.

Shams, L. and von der Malsburg, C. (1999). Are object shape primitives learnable? *Neurocomputing*, 26-27:855–863.
